# 3D Social Saliency from Head-mounted Cameras

**Hyun Soo Park**
Carnegie Mellon University
hyunsoop@cs.cmu.edu

**Eakta Jain**
Texas Instruments
e-jain@ti.com

**Yaser Sheikh**
Carnegie Mellon University
yaser@cs.cmu.edu

## Abstract

A gaze concurrence is a point in 3D where the gaze directions of two or more people intersect. It is a strong indicator of social saliency because the attention of the participating group is focused on that point. In scenes occupied by large groups of people, multiple concurrences may occur and transition over time. In this paper, we present a method to construct a 3D social saliency field and locate multiple gaze concurrences that occur in a social scene from videos taken by head-mounted cameras. We model the gaze as a cone-shaped distribution emanating from the center of the eyes, capturing the variation of eye-in-head motion. We calibrate the parameters of this distribution by exploiting the fixed relationship between the primary gaze ray and the head-mounted camera pose. The resulting gaze model enables us to build a social saliency field in 3D. We estimate the number and 3D locations of the gaze concurrences via provably convergent mode-seeking in the social saliency field. Our algorithm is applied to reconstruct multiple gaze concurrences in several real world scenes and evaluated quantitatively against motion-captured ground truth.

## 1 Introduction

Scene understanding approaches have largely focused on understanding the physical structure of a scene: "what is where?" [1]. In social scenes, i.e., scenes occupied by people, this definition of understanding needs to be expanded to include interpreting what is *socially salient* in that scene, such as who people interact with, where they look, and what they attend to. While classic structural scene understanding is an objective interpretation of the scene (e.g., 3D reconstruction [2], object recognition [3], or human affordance identification [4]), social scene understanding is subjective as it depends on the beholder and the particular group of people occupying the scene. For example, when we first enter a foyer during a party, we quickly look at different people and the groups they have formed, search for *personal* friends or acquaintances, and choose a group to join. Consider instead, an artificial agent, such as a social robot, that enters the same room: how should it interpret the social dynamics of the environment? The subjectivity of social environments makes the identification of quantifiable and measurable representations of social scenes difficult. In this paper, we aim to recover a representation of saliency in social scenes that approaches objectivity through the consensus of multiple subjective judgements.

Humans transmit visible social signals about what they find important and these signals are powerful cues for social scene understanding [5]. For instance, humans spontaneously orient their gaze to the target of their attention. When multiple people simultaneously pay attention to the same point in three dimensional space, e.g., at an obnoxious customer at a restaurant, their gaze rays[1] converge to a point that we refer to as a *gaze concurrence*. Gaze concurrences are foci of the 3D social saliency field of a scene. It is an effective approximation because although an individual's gaze indicates what he or she is subjectively interested in, a gaze concurrence encodes the consensus of multiple individuals. In a scene occupied by a larger number of people, multiple such concurrences may emerge as social cliques form and dissolve. In this paper, we present a method to reconstruct a 3D social saliency field and localize 3D gaze concurrences from videos taken by head-mounted cameras

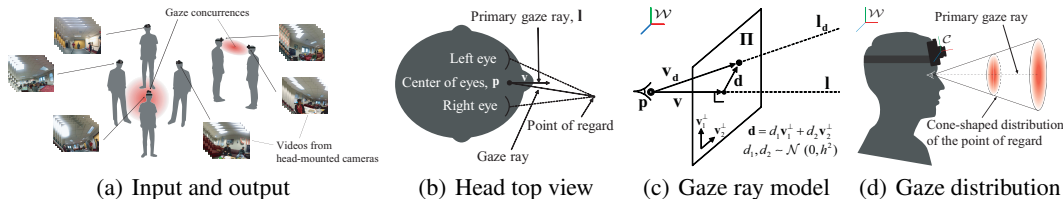

| (a) Input and output | (b) Head top view | (c) Gaze ray model | (d) Gaze distribution |

Figure 1: (a) In this paper, we present a method to reconstruct 3D gaze concurrences from videos taken by head-mounted cameras. (b) The primary gaze ray is a fixed 3D ray with respect to the head coordinate system and the gaze ray can be described by an angle with respect to the primary gaze ray. (c) The variation of the eye orientation is parameterized by a Gaussian distribution of the points on the plane, $\Pi$, which is normal to the primary gaze ray, $\mathbf{l}$ at unit distance from $\mathbf{p}$. (d) The gaze ray model results in a cone-shaped distribution of the point of regard.

on multiple people (Figure 1(a)). Our method automatically finds the number and location of gaze concurrences that may occur as people form social cliques in an environment.

**Why head-mounted cameras?** Estimating 3D gaze concurrences requires accurate estimates of the gaze of people who are widely distributed over the social space. For a third person camera, i.e., an outside camera looking into a scene, state-of-the-art face pose estimation algorithms cannot produce reliable face orientation and location estimation beyond approximately 45 degrees of a head facing the camera directly [6]. Furthermore, as they are usually fixed, third person views introduce spatial biases (i.e., head pose estimates would be better for people closer to and facing the camera) and limit the operating space. In contrast, head-mounted cameras instrument people rather than the scene. Therefore, one camera is used to estimate each head pose. As a result, 3D pose estimation of head-mounted cameras provides accurate and spatially unbiased estimates of the primary gaze ray[2].

Head-mounted cameras are poised to broadly enter our social spaces and many collaborative teams (such as search and rescue teams [8], police squads, military patrols, and surgery teams [9]) are already required to wear them. Head-mounted camera systems are increasingly becoming smaller, and will soon be seamlessly integrated into daily life [10].

**Contributions** The core contribution of this paper is an algorithm to estimate the 3D social saliency field of a scene and its modes from head-mounted cameras, as shown in Figure 1(a). This is enabled by a new model of gaze rays that represents the variation due to eye-in-head motion via a cone-shaped distribution. We present a novel method to calibrate the parameters of this model by leveraging the fact that the primary gaze ray is fixed with respect to the head-mounted camera in 3D. Given the collection of gaze ray distributions in 3D space, we automatically estimate the number and 3D locations of multiple gaze concurrences via mode-seeking in the social saliency field. We prove that the sequence of mode-seeking iterations converge. We evaluate our algorithm using motion capture data quantitatively, and apply it to real world scenes where social interactions frequently occur, such as meetings, parties, and theatrical performances.

## 2   Related Work

Humans transmit and respond to many different social signals when they interact with others. Among these signals, gaze direction is one of the most prominent visible signals because it usually indicates what the individual is interested in. In this context, gaze direction estimation has been widely studied in robotics, human-computer interaction, and computer vision [6, 11–22]. Gaze direction can be precisely estimated by the eye orientation. Wang and Sung [11] presented a system that estimates the direction of the iris circle from a single image using the geometry of the iris. Guestrin and Eizenman [12] and Hennessey and Lawrence [13] utilized corneal reflections and the vergence of the eye to infer the eye geometry and its motion, respectively. A head-mounted eye tracker is often used to determine the eye orientation [14, 15]. Although all these methods can estimate highly accurate gaze direction, either they can be used in a laboratory setting or the device occludes the viewer's field of view.

While the eyes are the primary source of gaze direction, Emery [16] notes that the head orientation is a strong indication of the direction of attention. For head orientation estimation, there are two approaches: outside-in and inside-out [23]. An outside-in system takes, as input, a third person image from a particular vantage point and estimates face orientation based on a face model. Murphy-Chutorian and Trivedi [6] have summarized this approach. Geometric modeling of the face has been used to orient the head by Gee and Cipolla [17] and Ballard and Stockman [18]. Rae and Ritter [19] estimated the head orientation via neural networks and Robertson and Reid [20] presented a method to estimate face orientation by learning 2D face features from different views in a low resolution video. With these approaches, a large number of cameras would need to be placed to cover a space large enough to contain all people. Also, the size of faces in these videos is often small, leading to biased head pose estimation depending on the distance from the camera. Instead of the outside-in approach, an inside-out approach estimates head orientation directly from a head-mounted camera looking out at the environment. Munn and Pelz [22] and Takemura et al. [15] estimated the head-mounted camera motion in 3D by feature tracking and visual SLAM, respectively. Pirri et al. [24] presented a gaze calibration procedure based on the eye geometry using 4 head-mounted cameras. We adopt an inside-out as it does not suffer from space limitations and biased estimation.

Gaze in a group setting has been used to identify social interaction or to measure social behavior. Stiefelhagen [25] and Smith et al. [26] estimated the point of interest in a meeting scene and a crowd scene, respectively. Bazzani et al. [27] introduced the 3D representation of the visual field of view, which enabled them to locate the convergence of views. Cristani et al. [28] adopted the F-formation concept that enumerates all possible spatial and orientation configurations of people to define the region of interest. However, these methods rely on data captured from the third person view point, i.e., outside-in systems and therefore, their capture space is limited and accuracy of head pose estimation degrades with distance from the camera. Our method is not subject to the same limitations. For an inside-out approach, Fathi et al. [29] present a method that uses a single first person camera to recognize discrete interactions within the wearer's immediate social clique. Their method is a complementary approach to our method as it analyzes the faces within a single person's field of view. In contrast, our approach analyzes an entire environment where several social cliques may form or dissolve over time.

## 3 Method

The videos from the head-mounted cameras are collected and reconstructed in 3D via structure from motion. Each person wears a camera on the head and performs a predefined motion for gaze ray calibration based on our gaze ray model (Section 3.1). After the calibration (Section 3.2), they may move freely and interact with other people. From the reconstructed camera poses in conjunction with the gaze ray model, we estimate multiple gaze concurrences in 3D via mode-seeking (Section 3.3).

Our camera pose registration in 3D is based on structure from motion as described in [2, 30, 31]. We first scan the area of interest (for example, the room or the auditorium) with a camera to reconstruct the reference structure. The 3D poses of the head-mounted cameras are recovered relative to the reference structure using a RANSAC [32] embedded Perspective-$n$-Point algorithm [33]. When some camera poses cannot be reconstructed because of lack of features or motion blur, we interpolate the missing camera poses based on the epipolar constraint between consecutive frames.

### 3.1 Gaze Ray Model

We represent the direction of the viewer's gaze as a 3D ray that is emitted from the center of the eyes and is directed towards the point of regard, as shown in Figure 1(b). The center of the eyes is fixed with respect to the head position and therefore, the orientation of the gaze ray in the world coordinate system is a composite of the head orientation and the eye orientation (eye-in-head motion). A head-mounted camera does not contain sufficient information to estimate the gaze ray because it can capture only the head position and orientation but not the eye orientation. However, when the motion of the point of regard is stabilized, i.e., when the point of regard is stationary or slowly moving with respect to the head pose, the eye orientation varies by a small degree [34–36] from the primary gaze ray. We represent the variation of the gaze ray with respect to the primary gaze ray by a Gaussian distribution on a plane normal to the primary gaze ray. The point of regard (and consequently, the gaze ray) is more likely to be near the primary gaze ray.

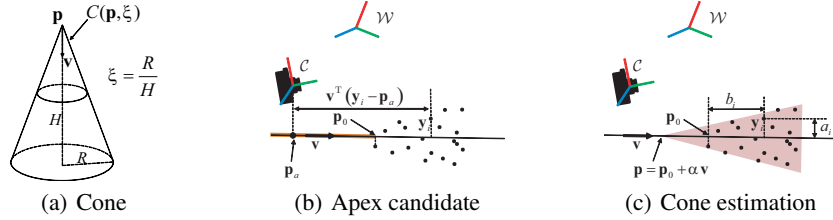

(a) Cone        (b) Apex candidate        (c) Cone estimation

Figure 2: (a) We parameterize our cone, $C$, with an apex, $\mathbf{p}$, and ratio, $\xi$, of the radius, $R$, to the height, $H$. (b) An apex can lie on the orange colored half line, i.e., behind $\mathbf{p}_0$. Otherwise some of the points are invisible. (c) An apex can be parameterized as $\mathbf{p} = \mathbf{p}_0 - \alpha\mathbf{v}$ where $\alpha > 0$. Equation (2) allows us to locate the apex accurately.

Let us define the primary gaze ray $\mathbf{l}$ by the center of the eyes $\mathbf{p} \in \mathbb{R}^3$, and the unit direction vector, $\mathbf{v} \in \mathbb{R}^3$ in the world coordinate system, $\mathcal{W}$, as shown in Figure 1(b). Any point on the primary gaze ray can be written as $\mathbf{p} + \alpha\mathbf{v}$ where $\alpha > 0$.

Let $\mathbf{\Pi}$ be a plane normal to the primary gaze ray $\mathbf{l}$ at unit distance from $\mathbf{p}$, as shown in Figure 1(c). The point $\mathbf{d}$ in $\mathbf{\Pi}$ can be written as $\mathbf{d} = d_1\mathbf{v}_1^{\perp} + d_2\mathbf{v}_2^{\perp}$ where $\mathbf{v}_1^{\perp}$ and $\mathbf{v}_2^{\perp}$ are two orthogonal vectors to $\mathbf{v}$ and $d_1$ and $d_2$ are scalars drawn from a Gaussian distribution, i.e., $d_1, d_2 \sim \mathcal{N}(0, h^2)$. This point $\mathbf{d}$ corresponds to the ray $\mathbf{l}_{\mathbf{d}}$ in 3D. Thus, the distribution of the points on the plane maps to the distribution of the gaze ray by parameterizing the 3D ray as $\mathbf{l}_{\mathbf{d}}(\mathbf{p}, \mathbf{v}_{\mathbf{d}}) = \mathbf{p} + \alpha\mathbf{v}_{\mathbf{d}}$ where $\mathbf{v}_{\mathbf{d}} = \mathbf{v} + \mathbf{d}$ and $\alpha > 0$. The resulting distribution of 3D points of regard is a cone-shaped distribution whose central axis is the primary gaze ray, i.e., a point distribution on any normal plane to the primary gaze ray is a scaled Gaussian centered at the intersection between $\mathbf{l}$ and the plane as shown in Figure 1(d).

## 3.2 Gaze Ray Calibration Algorithm

When a person wears a head-mounted camera, it may not be aligned with the direction of the primary gaze ray. In general, its center may not coincide with the center of the eyes either, as shown in Figure 1(d). The orientation and position offsets between the head-mounted camera and the primary gaze ray must be calibrated to estimate where the person is looking.

The relative transform between the primary gaze ray and the camera pose is constant across time because the camera is, for the most part, stationary with respect to the head, $\mathcal{C}$, as shown in Figure 1(d). Once the relative transform and camera pose have been estimated, the primary gaze ray can be recovered. We learn the primary gaze ray parameters, $\mathbf{p}$ and $\mathbf{v}$, with respect to the camera pose and the standard deviation $h$ of eye-in-head motion.

We ask people to form pairs and instruct each pair to look at each other's camera. While doing so, they are asked to move back and forth and side to side. Suppose two people A and B form a pair. If the cameras from A and B are temporally synchronized and reconstructed in 3D simultaneously, the camera center of B is the point of regard of A. Let $\mathbf{y}^{\mathcal{W}}$ (the camera center of B) be the point of regard of A and $\mathbf{R}$ and $\mathbf{C}$ be the camera orientation and the camera center of A, respectively. $\mathbf{y}^{\mathcal{W}}$ is represented in the world coordinate system, $\mathcal{W}$. We can transform $\mathbf{y}^{\mathcal{W}}$ to A's camera centered coordinate system, $\mathcal{C}$, by $\mathbf{y} = \mathbf{R}\mathbf{y}^{\mathcal{W}} - \mathbf{R}\mathbf{C}$. From $\{\mathbf{y}_i\}_{i=1,\cdots,n}$ where $n$ is the number of the points of regard, we can infer the primary gaze ray parameters with respect to the camera pose. If there is no eye-in-head motion, all $\{\mathbf{y}_i\}_{i=1,\cdots,n}$ will form a line which is the primary gaze ray. Due to the eye-in-head motion, $\{\mathbf{y}_i\}_{i=1,\cdots,n}$ will be contained in a cone whose central axis is the direction of the primary gaze ray, $\mathbf{v}$, and whose apex is the center of eyes, $\mathbf{p}$.

We first estimate the primary gaze line and then, find the center of the eye on the line to completely describe the primary gaze ray. To estimate the primary gaze line robustly, we embed line estimation by two points in the RANSAC framework [32][3]. This enables us to obtain a 3D line, $\mathbf{l}(\mathbf{p}_a, \mathbf{v})$ where $\mathbf{p}_a$ is the projection of the camera center onto the line and $\mathbf{v}$ is the direction vector of the line. The projections of $\{\mathbf{y}_i\}_{i=1,\cdots,n}$ onto the line will be distributed on a half line with respect to $\mathbf{p}_a$. This enables us to determine the sign of $\mathbf{v}$. Given this line, we find a 3D cone, $C(\mathbf{p}, \xi)$, that encapsulates

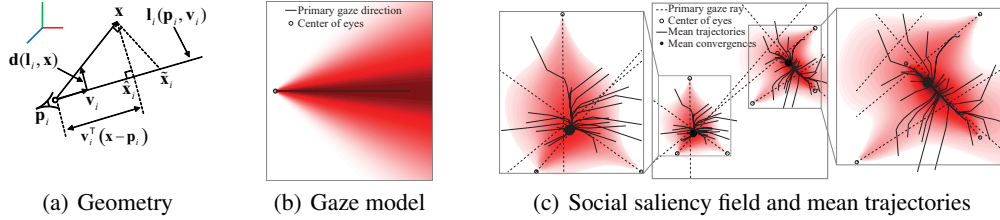

(a) Geometry      (b) Gaze model      (c) Social saliency field and mean trajectories

Figure 3: (a) $\widehat{\mathbf{x}}_i$ is the projection of $\mathbf{x}$ onto the primary gaze ray, $\mathbf{l}_i$, and $\mathbf{d}$ is a perspective distance vector defined in Equation (4). (b) Our gaze ray representation results in the cone-shaped distribution in 3D. (c) Two gaze concurrences are formed by seven gaze rays. High density is observed around the intersections of rays. Note that the maximum intensity projection is used to visualize the 3D density field. Our mean-shift algorithm allows any random points to converge to the highest density point accurately.

all $\{\mathbf{y}_i\}_{i=1,\cdots,n}$ where $\mathbf{p}$ is the apex and $\xi$ is the ratio of the radius, $R$, to height, $H$, as shown in Figure 2(a).

The apex can lie on a half line, which originates from the closest point, $\mathbf{p}_0$, to the center of the eyes and orients to $-\mathbf{v}$ direction, otherwise some $\mathbf{y}$ are invisible. In Figure 2(b), the apex must lie on the orange half line. $\mathbf{p}_0$ can be obtained as follows:

$$\mathbf{p}_0 = \mathbf{p}_a + \min\{\mathbf{v}^\mathsf{T}(\mathbf{y}_1 - \mathbf{p}_a), \cdots, \mathbf{v}^\mathsf{T}(\mathbf{y}_n - \mathbf{p}_a)\}\mathbf{v}. \tag{1}$$

Then, the apex can be written as $\mathbf{p} = \mathbf{p}_0 - \alpha\mathbf{v}$ where $\alpha > 0$, as shown in Figure 2(c).

There are an infinite number of cones which contain all points, e.g., any apex behind all points and $\xi = \infty$ can be a solution. Among these solutions, we want to find the tightest cone, where the minimum of $\xi$ is achieved. This also leads a degenerate solution where $\xi = 0$ and $\alpha = \infty$. We add a regularization term to avoid the $\alpha = \infty$ solution. The minimization can be written as,

$$\begin{aligned}
\underset{\alpha}{\text{minimize}} \quad & \xi + \lambda\alpha \\
\text{subject to} \quad & \frac{a_i}{b_i + \alpha} < \xi, \quad \forall\, i = 1, \cdots, n \\
& \alpha > 0
\end{aligned} \tag{2}$$

where $a_i = \left\|(\mathbf{I} - \mathbf{v}\mathbf{v}^\mathsf{T})(\mathbf{y}_i - \mathbf{p}_0)\right\|$ and $b_i = \mathbf{v}^\mathsf{T}(\mathbf{y}_i - \mathbf{p}_0)$ (Figure 2(c)), which are all known once $\mathbf{v}$ and $\mathbf{p}_0$ are known. $a_i/(b_i + \alpha) < \xi$ is the constraint that the cone encapsulates all points of regard $\{\mathbf{y}_i\}_{i=1,\cdots,n}$ and $\alpha > 0$ is the condition that the apex must be behind $\mathbf{p}_0$. $\lambda$ is a parameter that controls how far the apex is from $\mathbf{p}_0$. Equation (2) is a convex optimization problem (see Appendix in the supplementary material). Once the cone $C(\mathbf{p}, \xi)$ is estimated from $\{\mathbf{y}_i\}_{i=1,\cdots,n}$, $h$ is the standard deviation of the distance, $h = \mathtt{std}\{\|\mathbf{d}(\mathbf{l}, \mathbf{y}_i)\|\}_{i=1,\cdots,n}$, and will be used in Equation (3) as the bandwidth for the kernel density function.

### 3.3   Gaze Concurrence Estimation via Mode-seeking

3D gaze concurrences are formed at the intersections of multiple gaze rays, not at the intersection of multiple primary gazes (see Figure 1(b)). If we knew the 3D gaze rays, and which of rays shared a gaze concurrence, the point of intersection could be directly estimated via least squares estimation, for example. In our setup, neither one of these are known, nor do we know the number of gaze concurrences. With a head-mounted camera, only the primary gaze ray is computable; the eye-in-head motion is an unknown quantity. This precludes estimating the 3D gaze concurrence by finding a point of intersection, directly. In this section, we present a method to estimate the number and the 3D locations of gaze concurrences given primary gaze rays.

Our observations from head-mounted cameras are primary gaze rays. The gaze ray model discussed in Section 3.1 produces a distribution of points of regard for each primary gaze ray. The superposition of these distributions yields a 3D social saliency field. We seek modes in this saliency field via a mean-shift algorithm. The modes correspond to the gaze concurrences. The mean-shift algorithm [37] finds the modes by evaluating the weights between the current mean and observed points. We derive the closed form of the mean-shift vector directly from the observed primary gaze rays. While the observations are rays, the estimated modes are points in 3D. This formulation differs from the classic mean-shift algorithm where the observations and the modes lie in the same space.

For any point in 3D, $\mathbf{x} \in \mathbb{R}^3$, a density function (social saliency field), $f$, is generated by our gaze ray model. $f$ is the average of the Gaussian kernel density functions $K$ which evaluate the distance vector between the point, $\mathbf{x}$, and the primary gaze rays $\mathbf{l}_i$ as follows:

$$f(\mathbf{x}) = \frac{1}{N}\sum_{i=1}^{N} K\left(\frac{\mathbf{d}(\mathbf{l}_i,\mathbf{x})}{h_i}\right) = \frac{c}{N}\sum_{i=1}^{N}\frac{1}{h_i}k\left(\frac{\|\mathbf{d}(\mathbf{l}_i,\mathbf{x})\|^2}{h_i^2}\right) = \frac{1}{N}\sum_{i=1}^{N}\frac{1}{h_i\sqrt{2\pi}}\exp\left(-\frac{1}{2}\frac{\|\mathbf{d}(\mathbf{l}_i,\mathbf{x})\|^2}{h_i^2}\right), \quad (3)$$

where $N$ is the number of gaze rays and $h_i$ is a bandwidth set to be the standard deviation of eye-in-head motion obtained from the gaze ray calibration (Section 3.2) for the $i^{\text{th}}$ gaze ray. $k$ is the profile of the kernel density function, i.e., $K(\cdot) = ck(\|\cdot\|^2)/h$ and $c$ is a scaling constant. $\mathbf{d} \in \mathbb{R}^3$ is a perspective distance vector defined as

$$\mathbf{d}(\mathbf{l}_i(\mathbf{p}_i,\mathbf{v}_i),\mathbf{x}) = \begin{cases} \frac{\mathbf{x}-\widehat{\mathbf{x}}_i}{\mathbf{v}_i^\mathsf{T}(\mathbf{x}-\mathbf{p}_i)} & \text{for } \mathbf{v}_i^\mathsf{T}(\mathbf{x}-\mathbf{p}_i) \geq 0 \\ \infty & \text{otherwise,} \end{cases} \quad (4)$$

where $\widehat{\mathbf{x}}_i = \mathbf{p}_i + \mathbf{v}_i^\mathsf{T}(\mathbf{x}-\mathbf{p}_i)\mathbf{v}_i$, which is the projection of $\mathbf{x}$ onto the primary gaze ray as shown in Figure 3(a). $\mathbf{p}_i$ is the center of eyes and $\mathbf{v}_i$ is the direction vector for the $i^{\text{th}}$ primary gaze ray. Note that when $\mathbf{v}_i^\mathsf{T}(\mathbf{x}-\mathbf{p}_i) < 0$, the point is behind the eyes, and therefore is not visible. This distance vector directly captures the distance between $\mathbf{l}$ and $\mathbf{l_d}$ in the gaze ray model (Section 3.1) and therefore, this kernel density function yields a cone-shaped density field (Figure 1(d) and Figure 3(b)). Figure 3(c) shows a social saliency field (density field) generated by seven gaze rays. The regions of high density are the gaze concurrences. Note that the maximum intensity projection of the density field is used to illustrate a 3D density field.

The updated mean is the location where the maximum density increase can be achieved from the current mean. Thus, it moves along the gradient direction of the density function evaluated at the current mean. The gradient of the density function, $f(\mathbf{x})$, is

$$\nabla_{\mathbf{x}}f(\mathbf{x}) = \frac{2c}{N}\sum_{i=1}^{N}\frac{1}{h_i^3}k'\left(\left\|\frac{\mathbf{d}(\mathbf{l}_i,\mathbf{x})}{h_i}\right\|^2\right)\mathbf{d}(\mathbf{l}_i,\mathbf{x})^\mathsf{T}(\nabla_{\mathbf{x}}\mathbf{d}(\mathbf{l}_i,\mathbf{x})) = \frac{2c}{N}\left[\sum_{i=1}^{N}w_i\right]\left[\frac{\sum_{i=1}^{N}w_i\widetilde{\mathbf{x}}_i}{\sum_{i=1}^{N}w_i} - \mathbf{x}\right]^\mathsf{T}, \quad (5)$$

where

$$w_i = \frac{g\left(\left\|\frac{\mathbf{d}(\mathbf{l}_i,\mathbf{x})}{h_i}\right\|^2\right)}{h_i^3\left(\mathbf{v}_i^\mathsf{T}(\mathbf{x}-\mathbf{p}_i)\right)^2}, \quad \widetilde{\mathbf{x}}_i = \widehat{\mathbf{x}}_i + \frac{\|\mathbf{x}-\widehat{\mathbf{x}}_i\|^2}{\mathbf{v}_i^\mathsf{T}(\mathbf{x}-\mathbf{p})}\mathbf{v}_i,$$

and $g(x) = -k'(x)$. $\widetilde{\mathbf{x}}_i$ is the location that the gradient at $\mathbf{x}$ points to with respect to $\mathbf{l}_i$, as shown in Figure 3(a). Note that the gradient direction at $\mathbf{x}$ is perpendicular to the ray connecting $\mathbf{x}$ and $\mathbf{p}_i$. The last term of Equation (5) is the difference between the current mean estimate and the weighted mean. The new mean location, $\mathbf{x}^{j+1}$, can be achieved by adding the difference to the current mean estimate, $\mathbf{x}^j$:

$$\mathbf{x}^{j+1} = \frac{\sum_{i=1}^{N}w_i^j\widetilde{\mathbf{x}}_i^j}{\sum_{i=1}^{N}w_i^j}. \quad (6)$$

Figure 3(c) shows how our mean-shift vector moves random initial points according to the gradient information. The mean-shift algorithm always converges as shown in the following theorem.

**Theorem 1** *The sequence $\{f(\mathbf{x}^j)\}_{j=1,2,\cdots}$ provided by Equation (6) converges to the local maximum of the density field.*

See Appendix in the supplementary material for proof.

## 4  Result

We evaluate our algorithm quantitatively using a motion capture system to provide ground truth and apply it to real world examples where social interactions frequently occur. We use GoPro HD Hero2 cameras (www.gopro.com) and use the head mounting unit provided by GoPro. We synchronize the cameras using audio signals, e.g., a clap. In the calibration step, we ask people to form pairs, and move back and forth and side to side at least three times to allow the gaze ray model to be accurately estimated. For the initial points of the mean-shift algorithm, we sample several points on the primary gaze rays. This sampling results in convergences of the mean-shift because the local maxima form around the rays. If the weights of the estimated mode are dominated by only one gaze, we reject the mode, i.e., more than one gaze rays must contribute to estimate a gaze concurrence.

### 4.1 Validation with Motion Capture Data

We compare the 3D gaze concurrences estimated by our result with ground truth obtained from a motion capture system (capture volume: 8.3m×17.7m×4.3m). We attached several markers on a camera and reconstructed the camera motion using structure from motion and the motion capture system simultaneously. From the reconstructed camera trajectory, we recovered the similarity transform (scale, orientation, and translation) between two reconstructions. We placed two static markers and asked six people to move freely while looking at the markers. Therefore, the 3D gaze concurrences estimated by our algorithm should coincide with the 3D position of the static markers.

The top row in Figure 4(a) shows the trajectories of the gaze concurrences (solid lines) overlaid by the static marker positions (dotted lines). The mean error is 10.1cm with 5.73cm standard deviation. The bottom row in Figure 4(a) shows the gaze concurrences (orange and red points) with the ground truth positions (green and blue points) and the confidence regions (pink region) where a high value of the saliency field is achieved (region which has higher than 80% of the local maximum value). The ground truth locations are always inside these regions.

### 4.2 Real World Scenes

We apply our method to reconstruct 3D gaze concurrences in three real world scenes: a meeting, a musical, and a party. Figures 4(b), 5(a), and 5(b) show the reconstructed gaze concurrences and the projections of 3D gaze concurrences onto the head-mounted camera plane (top row). 3D renderings of the gaze concurrences (red dots) with the associated confidence region (salient region) are drawn in the middle row and the cone-shaped gaze ray models are also shown. The trajectories of the gaze concurrences are shown in the bottom row. The transparency of the trajectories encodes the timing.
**Meeting scene**: There were 11 people forming two groups: 6 for one group and 5 for the other group as shown in Figure 4(b). The people in each group started to discuss among themselves at the beginning (2 gaze concurrences). After a few minutes, all the people faced the presenter in the middle (50th frame: 1 gaze concurrence), and then they went back to their group to discuss again (445th frame: 2 gaze concurrences) as shown in Figure 4(b).
**Musical scene**: 7 audience members wore head-mounted cameras and watched the song, "Summer Nights" from the musical *Grease*. There were two groups of actors, "the pink ladies (women's group)" and "the T-birds (men's group)" and they sang the song alternatingly as shown in Figure 5(a). In the figure, we show the reconstruction of two frames when the pink ladies sang (41st frame) and when the T-birds sang (390th frame).
**Party scene**: there were 11 people forming 4 groups: 3 sat on couches, 3 talked to each other at the table, 3 played table tennis, and 2 played pool (178th frame: 4 gaze concurrences) as shown in Figure 5(b). Then, all moved to watch the table tennis game (710th frame: one gaze concurrence). Our method correctly evaluates the gaze concurrences at the location where people look. All results are best seen in the videos from the following project website (`http://www.cs.cmu.edu/~hyunsoop/gaze_concurrence.html`).

## 5 Discussion

In this paper, we present a novel representation for social scene understanding in terms of 3D gaze concurrences. We model individual gazes as a cone-shaped distribution that captures the variation of the eye-in-head motion. We reconstruct the head-mounted camera poses in 3D using structure from motion and estimate the relationship between the camera pose and the gaze ray. Our mode-seeking algorithm finds the multiple time-varying gaze concurrences in 3D. We show that our algorithm can accurately estimate the gaze concurrences.

When people's gaze rays are almost parallel, as in the musical scene (Figure 5(a)), the estimated gaze concurrences become poorly conditioned. The confidence region is stretched along the direction of the primary gaze rays. This is the case where the point of regard is very far away while people look at the point from almost the same vantage point. For such a scene, head-mounted cameras from different points of views can help to localize the gaze concurrences precisely.

Recognizing gaze concurrences is critical to collaborative activity. A future application of this work will be to use gaze concurrence to allow artificial agents, such as robots, to become collaborative team members that recognize and respond to social cues, rather than passive tools that require prompting. The ability to objectively measure gaze concurrences in 3D will also enable new investigations into social behavior, such as group dynamics, group hierarchies, and gender interactions, and

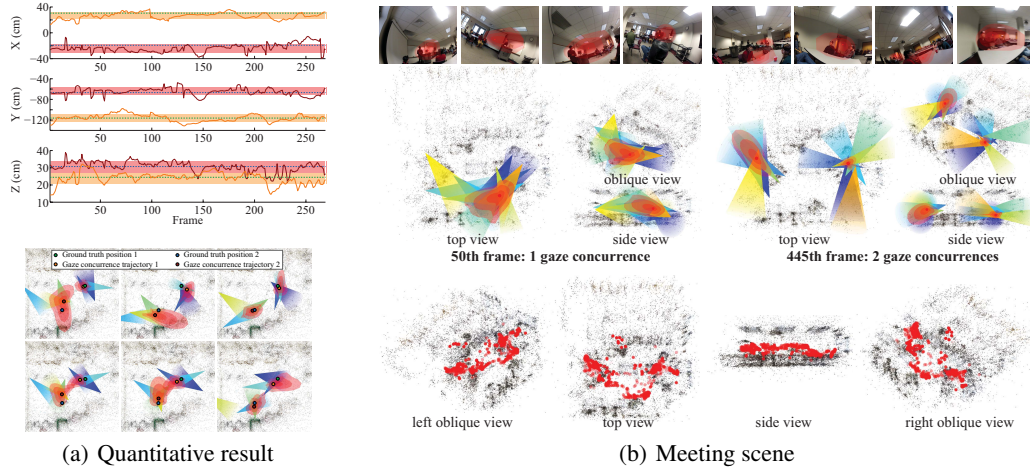

(a) Quantitative result       (b) Meeting scene

Figure 4: (a) Top: the solid lines (orange and red) are the trajectories of the gaze concurrences and the dotted lines (green and blue) are the ground truth marker positions. The colored bands are one standard deviation wide and are centered at the trajectory means. Bottom: there are two gaze concurrences with six people. (b) We reconstruct the gaze concurrences for the meeting scene. 11 head-mounted cameras were used to capture the scene. Top row: images with the reprojection of the gaze concurrences, middle row: rendering of the 3D gaze concurrences with cone-shaped gaze models, bottom row: the trajectories of the gaze concurrences.

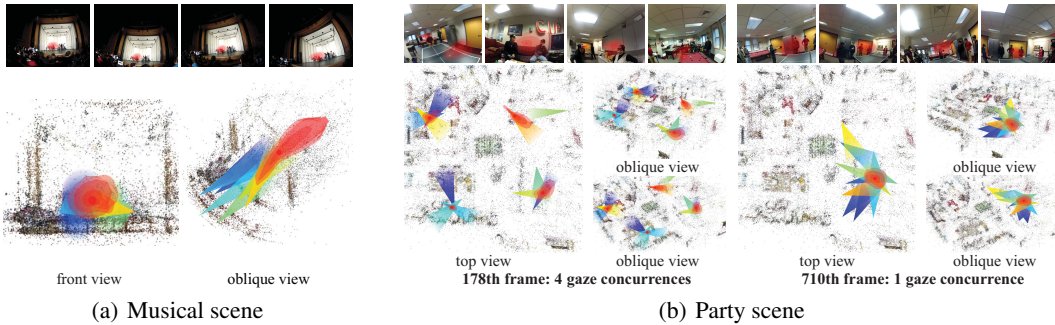

(a) Musical scene       (b) Party scene

Figure 5: (a) We reconstruct the gaze concurrences from musical audiences. 7 head-mounted cameras were used to capture the scene. (b) We reconstruct the gaze concurrences for the party scene. 11 head-mounted cameras were used to capture the scene. Top row: images with the reprojection of the gaze concurrences, bottom row: rendering of the 3D gaze concurrences with cone-shaped gaze models.

research into behavioral disorders, such as autism. We are interested in studying the spatiotemporal characteristics of the birth and death of gaze concurrences and how they relate to the groups in the scene.

## Acknowledgement

This work was supported by a Samsung Global Research Outreach Program, Intel ISTC-EC, NSF IIS 1029679, and NSF RI 0916272. We thank Jessica Hodgins, Irfan Essa, and Takeo Kanade for comments and suggestions on this work.

## Footnotes

[1]A gaze ray is a three dimensional ray emitted from the center of eyes and oriented to the point of regard as shown in Figure 1(b).

[2]The primary gaze ray is a fixed eye orientation with respect to the head. It has been shown that the orientation is a unique pose, independent of gravity, head posture, horizon, and the fusion reflex [7].

[3]We estimate a 3D line by randomly selecting two points at each iteration and find the line that produces the maximum number of inlier points.

## References

[1] D. Marr. *Vision: A Computational Investigation into the Human Representation and Processing of Visual Information*. Phenomenology and the Cognitive Sciences, 1982.

[2] N. Snavely, M. Seitz, and R. Szeliski. Photo tourism: Exploring photo collections in 3D. *TOG*, 2006.

[3] R. Fergus, P. Perona, and A. Zisserman. Object class recognition by unsupervised scale-invariant learning. In *CVPR*, 2003.

[4] A. Gupta, S. Satkin, A. A. Efros, and M. Hebert. From scene geometry to human workspace. In *CVPR*, 2011.

[5] A. Vinciarelli, M. Pantic, and H. Bourlard. Social signal processing: Survey of an emerging domain. *Image and Vision Computing*, 2009.

[6] E. Murphy-Chutorian and M. M. Trivedi. Head pose estimation in computer vision: A survey. *TPAMI*, 2009.

[7] R. S. Jampel and D. X. Shi. The primary position of the eyes, the resetting saccade, and the transverse visual head plane. head movements around the cervical joints. *Investigative Ophthalmology and Vision Science*, 1992.

[8] R. R. Murphy. Human-robot interaction in rescue robotics. *IEEE Trans. on Systems, Man and Cybernetics*, 2004.

[9] S. Marks, B. Wünsche, and J. Windsor. Enhancing virtual environment-based surgical teamwork training with non-verbal communication. In *GRAPP*, 2009.

[10] N. Bilton. A rose-colored view may come standard: Google glass. *The New York Times*, April 2012.

[11] J.-G. Wang and E. Sung. Study on eye gaze estimation. *IEEE Trans. on Systems, Man and Cybernetics*, 2002.

[12] E. D. Guestrin and M. Eizenman. General theory of remote gaze estimation using the pupil center and corneal reflection. *IEEE Trans. on Biomedical Engineering*, 2006.

[13] C. Hennessey and P. Lawrence. 3D point-of-gaze estimation on a volumetric display. In *ETRA*, 2008.

[14] D. Li, J. Babcock, and D. J. Parkhurst. openEyes: a low-cost head-mounted eye-tracking solution. In *ETRA*, 2006.

[15] K. Takemura, Y. Kohashi, T. Suenaga, J. Takamatsu, and T. Ogasawara. Estimating 3D point-of-regard and visualizing gaze trajectories under natural head movements. In *ETRA*, 2010.

[16] N. J. Emery. The eyes have it: the neuroethology, function and evolution of social gaze. *Neuroscience and Biobehavioral Reviews*, 2000.

[17] A. H. Gee and R. Cipolla. Determining the gaze of faces in images. *Image and Vision Computing*, 1994.

[18] P. Ballard and G. C. Stockman. Controlling a computer via facial aspect. *IEEE Trans. on Systems, Man and Cybernetics*, 1995.

[19] R. Rae and H. J. Ritter. Recognition of human head orientation based on artificial neural networks. *IEEE Trans. on Neural Networks*, 1998.

[20] N. M. Robertson and I. D. Reid. Estimating gaze direction from low-resolution faces in video. In *ECCV*, 2006.

[21] B. Noris, K. Benmachiche, and A. G. Billard. Calibration-free eye gaze direction detection with gaussian processes. In *GRAPP*, 2006.

[22] S. M. Munn and J. B. Pelz. 3D point-of-regard, position and head orientation from a portable monocular video-based eye tracker. In *ETRA*, 2008.

[23] G. Welch and E. Foxlin. Motion tracking: no silver bullet, but a respectable arsenal. *IEEE Computer Graphics and Applications*, 2002.

[24] F. Pirri, M. Pizzoli, and A. Rudi. A general method for the point of regard estimation in 3d space. In *CVPR*, 2011.

[25] R. Stiefelhagen, M. Finke, J. Yang, and A. Waibel. From gaze to focus of attention. In *VISUAL*, 1999.

[26] K. Smith, S. O. Ba, J.-M. Odobez, and D. Gatica-Perez. Tracking the visual focus of attention for a varying number of wandering people. *TPAMI*, 2008.

[27] L. Bazzani, D. Tosato, M. Cristani, M. Farenzena, G. Pagetti, G. Menegaz, and V. Murino. Social interactions by visual focus of attention in a three-dimensional environment. *Expert Systems*, 2011.

[28] M. Cristani, L. Bazzani, G. Paggetti, A. Fossati, D. Tosato, A. Del Bue, G. Menegaz, and V. Murino. Social interaction discovery by statistical analysis of F-formations. In *BMVC*, 2011.

[29] A. Fathi, J. K. Hodgins, and J. M. Rehg. Social interaction: A first-person perspective. In *CVPR*, 2012.

[30] R. I. Hartley and A. Zisserman. *Multiple View Geometry in Computer Vision*. Cambridge University Press, 2004.

[31] T. Shiratori, H. S. Park, L. Sigal, Y. Sheikh, and J. K. Hodgins. Motion capture from body-mounted cameras. *TOG*, 2011.

[32] M. A. Fischler and R. C. Bolles. Random sample consensus: a paradigm for model fitting with applications to image analysis and automated cartography. *Communications of the ACM*, 1981.

[33] V. Lepetit, F. Moreno-Noguer, and P. Fua. EPnP: An accurate O(n) solution to the PnP problem. *IJCV*, 2009.

[34] H. Misslisch, D. Tweed, and T. Vilis. Neural constraints on eye motion in human eye-head saccades. *Journal of Neurophysiology*, 1998.

[35] E. M. Klier, H. Wang, A. G. Constantin, and J. D. Crawford. Midbrain control of three-dimensional head orientation. *Science*, 2002.

[36] D. E. Angelaki and B. J. M. Hess. Control of eye orientation: where does the brain's role end and the muscle's begin? *European Journal of Neuroscience*, 2004.

[37] K. Fukunaga and L. D. Hostetler. The estimation of the gradient of a density function, with applications in pattern recognition. *IEEE Trans. on Information Theory*, 1975.

